# DUOL: A Double Updating Approach for Online Learning

**Peilin Zhao**
School of Comp. Eng.
Nanyang Tech. University
Singapore 639798
zhao0106@ntu.edu.sg

**Steven C.H. Hoi**
School of Comp. Eng.
Nanyang Tech. University
Singapore 639798
chhoi@ntu.edu.sg

**Rong Jin**
Dept. of Comp. Sci. & Eng.
Michigan State University
East Lansing, MI, 48824
rongjin@cse.msu.edu

## Abstract

In most online learning algorithms, the weights assigned to the misclassified examples (or support vectors) remain unchanged during the entire learning process. This is clearly insufficient since when a new misclassified example is added to the pool of support vectors, we generally expect it to affect the weights for the existing support vectors. In this paper, we propose a new online learning method, termed **Double Updating Online Learning**, or **DUOL** for short. Instead of only assigning a fixed weight to the misclassified example received in current trial, the proposed online learning algorithm also tries to update the weight for one of the existing support vectors. We show that the mistake bound can be significantly improved by the proposed online learning method. Encouraging experimental results show that the proposed technique is in general considerably more effective than the state-of-the-art online learning algorithms.

## 1   Introduction

Online learning has been extensively studied in the machine learning community (Rosenblatt, 1958; Freund & Schapire, 1999; Kivinen et al., 2001a; Crammer et al., 2006). Most online learning algorithms work by assigning a fixed weight to a new example when it is misclassified. As a result, the weights assigned to the misclassified examples, or support vectors, remain unchanged during the entire process of learning. This is clearly insufficient because when a new example is added to the pool of support vectors, we expect it to affect the weights assigned to the existing support vectors received in previous trials.

Although several online algorithms are capable of updating the example weights as the learning process goes, most of them are designed for the purposes other than improving the classification accuracy and reducing the mistake bound. For instance, in (Orabona et al., 2008; Crammer et al., 2003; Dekel et al., 2005), online learning algorithms are proposed to adjust the example weights in order to fit in the constraint of fixed number of support vectors; in (Cesa-Bianchi & Gentile, 2006), example weights are adjusted to track the drifting concepts. In this paper, we propose a new formulation for online learning that aims to dynamically update the example weights in order to improve the classification accuracy as well as the mistake bound. Instead of only assigning a weight to the misclassified example that is received in current trial, the proposed online learning algorithm also updates the weight for one of the existing support vectors. As a result, the example weights are dynamically updated as learning goes. We refer to the proposed approach as **Double Updating Online Learning**, or **DUOL** for short.

The key question in the proposed online learning approach is which one of the existing support vectors should be selected for weight updating. To this end, we employ an analysis for double updating online learning that is based on the recent work of online convex programming by incremental dual ascent (Shalev-Shwartz & Singer, 2006). Our analysis shows that under certain conditions, the proposed online learning algorithm can significantly reduce the mistake bound of the existing online algorithms. This result is further verified empirically by extensive experiments and comparison to the state-of-the-art algorithms for online learning.

The rest of this paper is organized as follows. Section 2 reviews the related work for online learning. Section 3 presents the proposed "double updating" approach to online learning. Section 4 gives our experimental results. Section 5 sets out the conclusion and addresses some future work.

## 2 Related Work

Online learning has been extensively studied in machine learning (Rosenblatt, 1958; Crammer & Singer, 2003; Cesa-Bianchi et al., 2004; Crammer et al., 2006; Fink et al., 2006; Yang et al., 2009). One of the most well-known online approaches is the Perceptron algorithm (Rosenblatt, 1958; Freund & Schapire, 1999), which updates the learning function by adding a new example with a constant weight into the current set of support vectors when it is misclassified. Recently a number of online learning algorithms have been developed based on the criterion of maximum margin (Crammer & Singer, 2003; Gentile, 2001; Kivinen et al., 2001b; Crammer et al., 2006; Li & Long, 1999). One example is the Relaxed Online Maximum Margin algorithm (ROMMA) (Li & Long, 1999), which repeatedly chooses the hyper-planes that correctly classify the existing training examples with the maximum margin. Another representative example is the Passive-Aggressive (PA) method (Crammer et al., 2006). It updates the classification function when a new example is misclassified or its classification score does not exceed some predefined margin. Empirical studies showed that the maximum margin based online learning algorithms are generally more effective than the Perceptron algorithm. However, despite the difference, most online learning algorithms only update the weight of the newly added support vector, and keep the weights of the existing support vectors unchanged. This constraint could significantly limit the effect of online learning.

Besides the studies for regular online learning, several algorithms are proposed for online learning with fixed budget. In these studies, the total number of support vectors is required to be bounded either by a theoretical bound or by a manually fixed budget. Example algorithms for fixed budget online learning include (Weston & Bordes, 2005; Crammer et al., 2003; Cavallanti et al., 2007; Dekel et al., 2008). The key idea of these algorithms is to dynamically update the weights of the existing support vectors as a new support vector is added, and the support vector with the least weight will be discarded when the number of support vectors exceeds the budget. The idea of discarding support vectors is also used in studies (Kivinen et al., 2001b) and (Cheng et al., 2006). In a very recently proposed method (Orabona et al., 2008), a new "projection" approach is proposed for online learning that ensures the number of support vectors is bounded. Besides, in (Cesa-Bianchi & Gentile, 2006), an online learning algorithm is proposed to handle the drifting concept, in which the weights of the existing support vectors are reduced whenever a new support vector is added. Although these online learning algorithms are capable of dynamically adjusting the weights of support vectors, they are designed to either fit in the budget of the number of support vectors or to handle drifting concepts, not to improve the classification accuracy and the mistake bound.

The proposed online learning algorithm is closely related to the recent work of online convex programming by incremental dual ascent (Shalev-Shwartz & Singer, 2006). Although the idea of simultaneously updating the weights of multiple support vectors was mentioned in (Shalev-Shwartz & Singer, 2006), no efficient updating algorithm was explicitly proposed. As will be shown later, the online algorithm proposed in this work shares the same computational cost as that of conventional online learning algorithms, despite the need of updating weights of two support vectors.

## 3 Double Updating to Online Learning

### 3.1 Motivation

We consider an online learning trial $t$ with an incoming example that is misclassified. Let $\kappa(\cdot, \cdot) : \mathbb{R}^d \times \mathbb{R}^d \to \mathbb{R}$ be the kernel function used in our classifier. Let $\mathcal{D} = \{(x_i, y_i), i = 1, \ldots, n\}$ be the collection of $n$ misclassified examples received before the trial $t$, where $x_i \in \mathbb{R}^d$ and $y_i \in \{-1, +1\}$. We also refer to these misclassified training examples as "support vectors". We denote by $\alpha = (\alpha_1, \ldots, \alpha_n) \in [0, C]^n$ the weights assigned to the support vectors in $\mathcal{D}$, where $C$ is a predefined constant. The resulting classifier, denoted by $f(x)$, is expressed as

$$f(x) = \sum_{i=1}^{n} \alpha_i y_i \kappa(x, x_i) \tag{1}$$

Let $(x_a, y_a)$ be the misclassified example received in the trial $t$, i.e., $y_a f(x_a) \leq 0$. In the conventional approach for online learning, we simply assign a constant weight, denoted by $\beta$, to $(x_a, y_a)$,

and the resulting classifier becomes

$$f'(x) = \beta y_a \kappa(x, x_a) + \sum_{i=1}^{n} \alpha_i y_i \kappa(x, x_i) = \beta y_a \kappa(x, x_a) + f(x) \tag{2}$$

The shortcoming with the conventional online learning approach is that the introduction of the new support vector $(x_a, y_a)$ may harm the classification of existing support vectors in $\mathcal{D}$, which is revealed by the following proposition.

**Proposition 1.** *Let $(x_a, y_a)$ be an example misclassified by the current classifier $f(x) = \sum_{i=1}^{n} \alpha_i y_i \kappa(x, x_i)$, i.e., $y_a f(x_a) < 0$. Let $f'(x) = \beta y_a \kappa(x, x_a) + f(x)$ be the updated classifier with $\beta > 0$. There exists at least one support vector $x_i \in \mathcal{D}$ such that $y_i f(x_i) > y_i f'(x_i)$.*

*Proof.* It follows from the fact that: $\exists x_i \in \mathcal{D}, y_i y_a \kappa(x_i, x_a) < 0$ when $y_a f(x_a) < 0$. □

As indicated by the above proposition, when a new misclassified example is added to the classifier, the classification confidence of at least one support vector will be reduced. In the case when $y_a f(x_a) \leq -\gamma$, it is easy to verify that there exists some support vector $(x_b, y_b)$ who satisfies $\beta y_a y_b k(x_a, x_b) \leq -\gamma/n$; at the meantime, it can be shown that when the classification confidence of $(x_b, y_b)$ is less than $\gamma/n$, i.e., $y_b f(x_b) \leq \gamma/n$, such support vector will be misclassified after the classifier is updated with the example $(x_a, y_a)$. In order to alleviate this problem, we propose to update the weight for the existing support vector whose classification confidence is significantly affected by the new misclassified example. In particular, we consider a support vector $(x_b, y_b) \in \mathcal{D}$ for weight updating if it satisfies the following two conditions

- $y_b f(x_b) \leq 0$, i.e., support vector $(x_b, y_b)$ is misclassified by the current classifier $f(x)$
- $k(x_b, x_a) y_a y_b \leq -\rho$ where $\rho \geq 0$ is a predefined threshold, i.e., support vector $(x_b, y_b)$ "**conflicts**" with the new misclassified example $(x_a, y_a)$.

We refer to the support vector satisfying the above conditions as **auxiliary example**. It is clear that by adding the misclassified example $(x_a, y_a)$ to classifier $f(x)$ with weight $\beta$, the classification score of $(x_b, y_b)$ will be reduced by at least $\beta\rho$, which could lead to the misclassification of the auxiliary example $(x_b, y_b)$. To avoid such a mistake, we propose to update the weights for both $(x_a, y_a)$ and $(x_b, y_b)$ simultaneously. In the next section, we show the details of the double updating algorithm for online learning, and the analysis for mistake bound.

Our analysis follows closely the previous work on the relationship between online learning and the dual formulation of SVM (Shalev-Shwartz & Singer, 2006), in which the online learning is interpreted as an efficient updating rule for maximizing the objective function in the dual form of SVM. We denote by $\Delta_t$ the improvement of the objective function in dual SVM when adding a new misclassified example to the classification function in the $t$-th trial. If an online learning algorithm $\mathcal{A}$ is designed to ensure that all $\Delta_t$ is bounded from the below by a positive constant $\Delta$, then the number of mistakes made by $\mathcal{A}$ when trained over a sequence of trials $(x_1, y_1), \ldots, (x_T, y_T)$, denoted by $M$, is upper bounded by:

$$M \leq \frac{1}{\Delta} \left( \min_{f \in \mathcal{H}_\kappa} \frac{1}{2} \|f\|_{\mathcal{H}_\kappa}^2 + C \sum_{i=1}^{T} \ell(y_i f(x_i)) \right) \tag{3}$$

where $\ell(y_i f(x_i)) = \max(0, 1 - y_i f(x_i))$ is the hinge loss function. In our analysis, we will show that $\Delta$, which is referred to as the **bounding constant** for the improvement in the objective function, could be significantly improved when updating the weight for both the newly misclassified example and the auxiliary example.

For the remaining part of this paper, we denote by $(x_b, y_b)$ an auxiliary example that satisfies the two conditions specified before. We slightly abuse the notation by using $\alpha = (\alpha_1, \ldots, \alpha_{n-1})) \in \mathbb{R}^{n-1}$ to denote the weights assigned to all the support vectors in $\mathcal{D}$ except $(x_b, y_b)$. Similarly, we denote by $\mathbf{y} = (y_1, \ldots, y_{n-1}) \in [-1, 1]^{n-1}$ the class labels assigned to all the examples in $\mathcal{D}$ except for $(x_b, y_b)$. We define

$$s_a = \kappa(x_a, x_a), \quad s_b = \kappa(x_b, x_b), \quad s_{ab} = \kappa(x_a, x_b), \quad w_{ab} = y_a y_b s_{ab}. \tag{4}$$

According to the assumption of auxiliary example, we have $w_{ab} = s_{ab} y_a y_b \leq -\rho$. Finally, we denote by $\widehat{\gamma_b}$ the weight for the auxiliary example $(x_b, y_b)$ that is used in the current classifier $f(x)$, and by $\gamma_a$ and $\gamma_b$ the updated weights for $(x_a, y_a)$ and $(x_b, y_b)$, respectively. Throughout the analysis, we assume $\kappa(x, x) \leq 1$ for any example $x$.

## 3.2 Double Updating Online Learning

Recall an auxiliary example $(x_b, y_b)$ should satisfy two conditions (I) $y_b f(x_b) \leq 0$, and (II) $w_{ab} \leq -\rho$. In addition, the new example $(x_a, y_a)$ received in the current iteration $t$ is misclassified, i.e., $y_a f(x_a) \leq 0$. Following the framework of dual formulation for online learning, the following lemma shows how to compute $\Delta_t$, i.e., the improvement in the objective function of dual SVM by adjusting weights for $(x_a, y_a)$ and $(x_b, y_b)$.

**Lemma 1.** *The maximal improvement in the objective function of dual SVM by adjusting weights for $(x_a, y_a)$ and $(x_b, y_b)$, denoted by $\Delta_t$, is computed by solving the following optimization problem:*

$$\Delta_t = \max_{\gamma_a, \Delta\gamma_b} \{h(\gamma_a, \Delta\gamma_b) : 0 \leq \gamma_a \leq C, \ 0 \leq \Delta\gamma_b \leq C - \widehat{\gamma}_b\} \tag{5}$$

*where*

$$h(\gamma_a, \Delta\gamma_b) = \gamma_a(1 - y_a f(x_a)) + \Delta\gamma_b(1 - y_b f(x_b)) - \frac{s_a}{2}\gamma_a^2 - \frac{s_b}{2}\Delta\gamma_b^2 - w_{ab}\gamma_a\Delta\gamma_b \tag{6}$$

*Proof.* It is straightforward to verify that the dual function of $\min_{f_t \in \mathcal{H}_\kappa} \frac{1}{2}\|f_t\|_{\mathcal{H}_\kappa}^2 + C \sum_{i=1}^{t} \ell(y_i f_t(x_i))$, denoted by $D_t(\gamma_1, \ldots, \gamma_t)$, is computed as follows,

$$D_t(\gamma_1, \ldots, \gamma_t) = \sum_{i=1}^{t} \gamma_i - \sum_{i=1}^{t} \gamma_i y_i f_t(x_i) + \frac{1}{2}\|f_t\|_{H_\kappa}^2 \tag{7}$$

where $0 \leq \gamma_i \leq C$, $i = 1, \ldots, t$ and $f_t(\cdot) = \sum_{i=1}^{t} \gamma_i y_i \kappa(\cdot, x_i)$ is the current classifier. Thus,

$$\begin{aligned}
h(\gamma_a, \Delta\gamma_b) &= D_t(\gamma_1, \ldots, \widehat{\gamma}_b + \Delta\gamma_b, \ldots, \gamma_{t-1}, \gamma_a) - D_{t-1}(\gamma_1, \ldots, \widehat{\gamma}_b, \ldots, \gamma_{t-1}) \\
&= \sum_{i=1}^{t-1} \gamma_i + \Delta\gamma_b + \gamma_a - \left(\sum_{i=1}^{t-1} \gamma_i y_i f_t(x_i) + \Delta\gamma_b y_b f_t(x_b) + \gamma_a y_a f_t(x_a)\right) + \frac{1}{2}\|f_t\|_{H_\kappa}^2 \\
&\quad - \left(\sum_{i=1}^{t-1} \gamma_i - \sum_{i=1}^{t-1} \gamma_i y_i f_{t-1}(x_i) + \frac{1}{2}\|f_{t-1}\|_{H_\kappa}^2\right)
\end{aligned}$$

Using the relation $f_t(x) = f_{t-1}(x) + \Delta\gamma_b y_b \kappa(x, x_b) + \gamma_a y_a \kappa(x, x_a)$, we have

$$h(\gamma_a, \Delta\gamma_b) = \gamma_a(1 - y_a f_{t-1}(x_a)) + \Delta\gamma_b(1 - y_b f_{t-1}(x_b)) - \frac{s_a}{2}\gamma_a^2 - \frac{s_b}{2}\Delta\gamma_b^2 - w_{ab}\gamma_a\Delta\gamma_b$$

Finally, we need to show $\Delta\gamma_b \geq 0$. Note that this constraint does not come directly from the box constraint that the weight for example $(x_b, y_b)$ is in the range $[0, C]$, i.e., $\widehat{\gamma}_b + \Delta\gamma_b \in [0, C]$. To this end, we consider the part of $h(\gamma_a, \Delta\gamma_b)$ that is related to $\Delta\gamma_b$, i.e.,

$$g(\Delta\gamma_b) = \Delta\gamma_b(1 - y_b f_{t-1}(x_b) - w_{ab}\gamma_a) - \frac{s_b}{2}\Delta\gamma_b^2$$

Since $w_{ab} \leq -\rho$ and $y_b f_{t-1}(x_b) \leq 0$, it is clear that $\Delta\gamma_b \geq 0$ when maximizing $g(\Delta\gamma_b)$, which results in the constraint $\Delta\gamma_b \geq 0$. $\qquad\square$

The following theorem shows the bound for $\Delta$ when $C$ is sufficiently large.

**Theorem 1.** *Assume $C > \widehat{\gamma}_b + 1/(1 - \rho)$ for the selected auxiliary example $(x_b, y_b)$. We have the following bound for $\Delta$*

$$\Delta \geq \frac{1}{1 - \rho} \tag{8}$$

*Proof.* Using the fact $s_a, s_b \leq 1$, $\gamma_a, \Delta\gamma_b \geq 0$, $y_a f(x_a) \leq 0$, $y_b f(x_b) \leq 0$, and $w_{a,b} \leq -\rho$, we have

$$h(\gamma_a, \Delta\gamma_b) \geq \gamma_a + \Delta\gamma_b - \frac{1}{2}\gamma_a^2 - \frac{1}{2}\Delta\gamma_b^2 + \rho\gamma_a\Delta\gamma_b$$

Thus, $\Delta$ is bounded as

$$\Delta \geq \max_{\gamma_b \in [0,C], \Delta\gamma_b \in [0, C-\widehat{\gamma}]} \gamma_a + \Delta\gamma_b - \frac{1}{2}(\gamma_a^2 + \Delta\gamma_b^2) + \rho\gamma_a\Delta\gamma_b$$

Under the condition that $C > \hat{\gamma}_b + 1/(1 - \rho)$, it is easy to verify that the optimal solution for the above problem is $\gamma_a = \Delta\gamma_b = 1/(1 - \rho)$, which leads to the result in the theorem. $\qquad\square$

We now consider the general case, where we only assume $C \geq 1$. The following theorem shows the bound for $\Delta$ in the general case.

**Theorem 2.** *Assume $C \geq 1$. We have the following bound for $\Delta$, when updating the weights for the new example $(x_a, y_a)$ and the auxiliary example $(x_b, y_b)$*

$$\Delta \geq \frac{1}{2} + \frac{1}{2} \min \left( (1 + \rho)^2, (C - \widehat{\gamma})^2 \right)$$

*Proof.* By setting $\gamma_a = 1$, we have $h(\gamma_a, \Delta\gamma_b)$ computed as

$$h(\gamma_a = 1, \Delta\gamma_b) \geq \frac{1}{2} + (1 + \rho)\Delta\gamma_b - \frac{1}{2}\Delta\gamma_b^2$$

Hence, $\Delta$ is lower bounded by

$$\Delta \geq \frac{1}{2} + \max_{\Delta\gamma_b \in [0, C - \widehat{\gamma}]} \left( (1 + \rho)\Delta\gamma_b - \frac{1}{2}\Delta\gamma_b^2 \right) \geq \frac{1}{2} + \frac{1}{2} \min \left( (1 + \rho)^2, (C - \widehat{\gamma})^2 \right)$$

$\square$

Since we only have $\Delta \geq 1/2$ if we only update the weight for the new misclassified example $(x_a, y_a)$, the result in theorem 2 indicates an increase in $\Delta$ when updating the weight for both $(x_a, y_a)$ and the auxiliary example $(x_b, y_b)$. Furthermore, when $C$ is sufficiently large, as indicated by Theorem 1, the improvement in $\Delta$ can be very significant.

The final remaining question is how to identify the auxiliary example $(x_b, y_b)$ efficiently, which requires efficiently updating the classification score $y_i f(x_i)$ for all the support vectors. To this end, we introduce a variable for each support vector, denoted by $f_t^i$, to keep track the classification score. When a new support vector $(x_a, y_a)$ with weight $\gamma_a$ is added to the classifier, we update the classification score $f_{t-1}^i$ by $f_t^i \leftarrow f_{t-1}^i + y_i \gamma_a y_a \kappa(x_i, x_a)$, and when the weight of an auxiliary example $(x_b, y_b)$ is updated from $\hat{\gamma}_b$ to $\gamma_b$, we update the classification score $f_{t-1}^i$ by $f_t^i \leftarrow f_{t-1}^i + y_i(\gamma_b - \hat{\gamma}_b)y_b \kappa(x_i, x_b)$. This updating procedure ensures that the computational cost of double updating online learning is $O(n)$, where $n$ is the number of support vectors, similar to that of the kernel online learning algorithm. Figure 1 shows the details of the DUOL algorithm.

Finally, we show a bound on the number of mistakes by assuming $C$ is sufficiently large.

**Theorem 3.** *Let $(x_1, y_1), \ldots, (x_T, y_T)$ be a sequence of examples, where $x_t \in \mathbb{R}^n$, $y_t \in \{-1, +1\}$ and $\kappa(x_t, x_t) \leq 1$ for all $t$. And assume $C$ is sufficiently large. Then for any function $f$ in $\mathcal{H}_\kappa$, the number of prediction mistakes $M$ made by DUOL on this sequence of examples is bounded by:*

$$M \leq 2 \left( \min_{f \in \mathcal{H}_\kappa} \frac{1}{2} \|f\|_{\mathcal{H}_\kappa}^2 + C \sum_{i=1}^{T} \ell(y_i f(x_i)) \right) - \frac{1 + \rho}{1 - \rho} M_d(\rho) \tag{9}$$

*where $M_d(\rho)$ is the number of mistakes when there is an auxiliary example, which depends on the threshold $\rho$ and the dataset ($M_d(\rho)$ is actually a decreasing function with $\rho$).*

*Proof.* We denote by $M_s$ the number of mistakes when we made a single update without finding appropriate auxiliary example. Using Theorem 1, we have the following inequality,

$$\frac{1}{2} M_s + \frac{1}{1 - \rho} M_d(\rho) \leq \left( \min_{f \in \mathcal{H}_\kappa} \frac{1}{2} \|f\|_{\mathcal{H}_\kappa}^2 + C \sum_{i=1}^{T} \ell(y_i f(x_i)) \right) \tag{10}$$

Plugging $M = M_s + M_d$ into the equation above, we can get

$$M \leq 2 \left( \min_{f \in \mathcal{H}_\kappa} \frac{1}{2} \|f\|_{\mathcal{H}_\kappa}^2 + C \sum_{i=1}^{T} \ell(y_i f(x_i)) \right) - \frac{1 + \rho}{1 - \rho} M_d(\rho) \tag{11}$$

$\square$

It is worthwhile pointing out that although according to Theorem 3, it seems that the larger the value of $\rho$ the smaller the mistake bound will be. This however is not true since $M_d(\rho)$ is in general a monotonically decreasing function in $\rho$. As a result, it is unclear if $M_d(\rho) \times (1 + \rho)/(1 - \rho)$ will increase when $\rho$ is increased.

| | |
|---|---|
| **Algorithm 1** The DUOL Algorithm (**DUOL**)<br>PROCEDURE<br>1:   Initialize $S_0 = \emptyset$, $f_0 = 0$;<br>2:   **for** t=1,2,...,T **do**<br>3:       Receive new instance $\mathbf{x}_t$<br>4:       Predict $\hat{y}_t = \text{sign}(f_{t-1}(\mathbf{x}_t))$;<br>5:       Receive label $y_t$;<br>6:       $l_t = max\{0, 1 - y_t f_{t-1}(\mathbf{x}_t)\}$<br>7:       **if** $l_t > 0$ **then**<br>8:           $w_{min} = 0$<br>9:           **for** $\forall i \in S_{t-1}$ **do**<br>10:              **if** $(f_{t-1}^i \leq 0)$ **then**<br>11:                  **if** $(y_i y_t k(\mathbf{x}_i, \mathbf{x}_t) < w_{min})$ **then**<br>12:                      $w_{min} = y_i y_t k(\mathbf{x}_i, \mathbf{x}_t)$;<br>13:                      $(\mathbf{x}_b, y_b) = (\mathbf{x}_i, y_i)$;/*auxiliary example*/<br>14:                  **end if**<br>15:              **end if**<br>16:          **end for**<br>17:          $f_{t-1}^t = y_t f_{t-1}(\mathbf{x}_t)$;<br>18:          $S_t = S_{t-1} \cup \{t\}$;<br>19:          **if** $(w_{min} \leq -\rho)$ **then** | 20:              $\gamma_t = min(C, \frac{1}{1-\rho})$;<br>21:              $\gamma_b = min(C, \hat{\gamma}_b + \frac{1}{1-\rho})$;<br>22:              **for** $\forall i \in S_t$ **do**<br>23:                  $f_t^i \leftarrow f_{t-1}^i + y_i \gamma_t y_t k(\mathbf{x}_i, \mathbf{x}_t)$<br>                        $+ y_i(\gamma_b - \hat{\gamma}_b)y_b k(\mathbf{x}_i, \mathbf{x}_b)$;<br>24:              **end for**<br>25:              $f_t = f_{t-1} + \gamma_t y_t k(\mathbf{x}_t, \cdot) + (\gamma_b - \hat{\gamma}_b)y_b k(\mathbf{x}_b, \cdot)$;<br>26:          **else** /* no auxiliary example found */<br>27:              $\gamma_t = min(C, 1)$;<br>28:              **for** $\forall i \in S_t$ **do**<br>29:                  $f_t^i \leftarrow f_{t-1}^i + y_i \gamma_t y_t k(\mathbf{x}_i, \mathbf{x}_t)$;<br>30:              **end for**<br>31:              $f_t = f_{t-1} + \gamma_t y_t k(\mathbf{x}_t, \cdot)$;<br>32:          **end if**<br>33:      **else**<br>34:          $f_t = f_{t-1}$; $S_t = S_{t-1}$;<br>35:          **for** $\forall i \in S_t$ **do**<br>36:              $f_t^i \leftarrow f_{t-1}^i$;<br>37:          **end for**<br>38:      **end if**<br>39:  **end for** |

Figure 1: The Algorithm of Double Updating Online Learning (DUOL).

# 4 Experimental Results

## 4.1 Experimental Testbed and Setup

We now evaluate the empirical performance of the proposed double updating online learning (DUOL) algorithm. We compare DUOL with a number of state-of-the-art techniques, including Perceptron (Rosenblatt, 1958; Freund & Schapire, 1999), the "ROMMA" algorithm and its aggressive version "agg-ROMMA" (Li & Long, 1999), the $ALMA_p(\alpha)$ algorithm (Gentile, 2001), and the Passive-Aggressive algorithms ("PA") (Crammer et al., 2006). The original Perceptron algorithm was proposed for learning linear models. In our experiments, we follow (Kivinen et al., 2001b) by adapting it to the kernel case. Two versions of PA algorithms (PA-I and PA-II) were implemented as described in (Crammer et al., 2006). Finally, as an ideal yardstick, we also implement a full online SVM algorithm ("Online-SVM") (Shalev-Shwartz & Singer, 2006), which updates all the support vectors in each trial, and is thus computationally extremely intensive as will be revealed in our study.

To extensively examine the performance, we test all the algorithms on a number of benchmark datasets from web machine learning repositories. All of the datasets can be downloaded from LIB-SVM website [1], UCI machine learning repository [2] and MIT CBCL face datasets [3] . Due to space limitation, we randomly choose six of them in our discussions, including "german", "splice", "spambase", "MITFace", "a7a", and "w7a".

To make a fair comparison, all algorithms adopt the same experimental setup. In particular, for all the compared algorithms, we set the penalty parameter $C = 5$, and employ the same Gaussian kernel with $\sigma = 8$. For the $ALMA_p(\alpha)$ algorithm, parameter $p$ and $\alpha$ are set to be 2 and 0.9, respectively, based on our experience. For the proposed DUOL algorithm, we fix $\rho$ to be 0.2 for all cases.

All the experiments were conducted over 20 random permutations for each dataset. All the results were reported by averaging over these 20 runs. We evaluate the online learning performance by measuring *mistake rate*, i.e., the ratio of the number of mistakes made by the online learning algorithm over the total number of examples received for predictions. In addition, to examine the sparsity of the resulting classifiers, we also evaluate the number of support vectors produced by each online learning algorithm. Finally, we also evaluate computational efficiency of all the algorithms by their running time (in seconds). All experiments were run in Matlab over a machine of 2.3GHz CPU.

## 4.2 Performance Evaluation

Table 1 to 6 summarize the performance of all the compared algorithms over the six datasets[4], respectively. Figure 2 to 6 show the mistake rates of all online learning algorithms in comparison over trials. We observe that Online-SVM yields considerably better performance than the other online learning algorithms for dataset "german", "splice", "spambase", and "MITFace", however, at the price of extremely high computational cost. For most cases, the running time of Online-SVM is two order, sometimes three order, higher than the other online learning algorithms, making it

unsuitable for online learning. For the remaining part of this section, we restrict our discussion to the other six baseline online learning algorithms.

First, among the six baseline algorithms in comparison, we observe that the agg-ROMMA and two PA algorithms (PA-I and PA-II) perform considerably better than the other three algorithms (i.e., Perceptron, ROMMA, and ALMA) in most cases. We also notice that the agg-ROMMA and the two PA algorithms consume considerably larger numbers of support vectors than the other three algorithms. We believe this is because the agg-ROMMA and the two PA algorithms adopt more aggressive strategies than the other three algorithms, resulting more updates and better classification performance. For the convenience of discussion, we refer to agg-ROMMA and two PA algorithms as aggressive algorithms, and the three algorithms as non-aggressive ones.

Second, comparing with all six competing algorithms, we observe that DUOL achieves significantly smaller *mistake rates* than the other single-updating algorithms in all cases. This shows that the proposed double updating approach is effective in improving the online prediction performance. By examining the sparsity of resulting classifiers, we observed that DUOL results in sparser classifiers than the three aggressive online learning algorithms, and denser classifiers than the three non-aggressive algorithms.

Third, according to the results of running time, we observe that DUOL is overall efficient compared to the state-of-the-art online learning algorithms. Among all the compared algorithms, Perceptron, for its simplicity, is clearly the most efficient algorithm, and the agg-ROMMA algorithm is significantly slower than the others (except for "Online-SVM"). Although DUOL requires double updating, its efficiency is comparable to the PA and ROMMA algorithms.

Table 1: Evaluation on *german* (n=1000, d=24).

| Algorithm | Mistake (%) | Support Vectors (#) | Time (s) |
|---|---|---|---|
| Perceptron | 35.305 ± 1.510 | 353.05 ± 15.10 | 0.018 |
| ROMMA | 35.105 ± 1.189 | 351.05 ± 11.89 | 0.154 |
| agg-ROMMA | 33.350 ± 1.287 | 643.25 ± 12.31 | 1.068 |
| ALMA$_2$(0.9) | 34.025 ± 0.910 | 402.00 ± 7.33 | 0.225 |
| PA-I | 33.670 ± 1.278 | 732.60 ± 9.74 | 0.029 |
| PA-II | 33.175 ± 1.229 | 757.00 ± 10.02 | 0.030 |
| Online-SVM | 28.860 ± 0.651 | 646.10 ± 5.00 | 16.097 |
| DUOL | 29.990 ± 1.033 | 682.50 ± 12.87 | 0.089 |

Table 2: Evaluation on *splice* (n=1000, d=6).

| Algorithm | Mistakes (%) | Support Vectors (#) | Time (s) |
|---|---|---|---|
| Perceptron | 27.120 ± 0.975 | 271.20 ± 9.75 | 0.016 |
| ROMMA | 25.560 ± 0.814 | 255.60 ± 8.14 | 0.055 |
| agg-ROMMA | 22.980 ± 0.780 | 602.95 ± 7.43 | 0.803 |
| ALMA$_2$(0.9) | 26.040 ± 0.965 | 314.95 ± 9.41 | 0.075 |
| PA-I | 23.815 ± 1.042 | 665.60 ± 5.60 | 0.028 |
| PA-II | 23.515 ± 1.005 | 689.00 ± 7.85 | 0.028 |
| Online-SVM | 17.455 ± 0.518 | 614.90 ± 2.92 | 12.243 |
| DUOL | 20.560 ± 0.566 | 577.85 ± 8.93 | 0.076 |

Table 3: Evaluation on *spambase* (n=4601, d=57).

| Algorithm | Mistake (%) | Support Vectors (#) | Time (s) |
|---|---|---|---|
| Perceptron | 24.987 ± 0.525 | 1149.65 ± 24.17 | 0.204 |
| ROMMA | 23.953 ± 0.510 | 1102.10 ± 23.44 | 10.128 |
| agg-ROMMA | 21.242 ± 0.384 | 2550.60 ± 27.32 | 95.028 |
| ALMA$_2$(0.9) | 23.579 ± 0.411 | 1550.15 ± 15.65 | 25.294 |
| PA-I | 22.112 ± 0.374 | 2861.50 ± 24.36 | 0.490 |
| PA-II | 21.907 ± 0.340 | 3029.10 ± 24.69 | 0.505 |
| Online-SVM | 17.138 ± 0.321 | 2396.95 ± 10.57 | 2521.665 |
| DUOL | 19.438 ± 0.432 | 2528.55 ± 20.57 | 0.985 |

Table 4: Evaluation on *MITFace* (n=6977, d=361).

| Algorithm | Mistake (%) | Support Vectors (#) | Time (s) |
|---|---|---|---|
| Perceptron | 4.665 ± 0.192 | 325.50 ± 13.37 | 0.164 |
| ROMMA | 4.114 ± 0.155 | 287.05 ± 10.84 | 0.362 |
| agg-ROMMA | 3.137 ± 0.093 | 1121.15 ± 24.18 | 11.074 |
| ALMA$_2$(0.9) | 4.467 ± 0.169 | 400.10 ± 10.53 | 0.675 |
| PA-I | 3.190 ± 0.128 | 1155.45 ± 14.53 | 0.356 |
| PA-II | 3.108 ± 0.112 | 1222.05 ± 13.73 | 0.370 |
| Online-SVM | 1.142 ± 0.073 | 520.05 ± 4.55 | 7238.105 |
| DUOL | 2.409 ± 0.161 | 768.65 ± 16.18 | 0.384 |

Table 5: Evaluation on *a7a* (n=16100, d=123).

| Algorithm | Mistake (%) | Support Vectors (#) | Time (s) |
|---|---|---|---|
| Perceptron | 22.022 ± 0.202 | 3545.50 ± 32.49 | 2.043 |
| ROMMA | 21.297 ± 0.272 | 3428.85 ± 43.77 | 306.793 |
| agg-ROMMA | 20.832 ± 0.234 | 4541.30 ± 109.39 | 661.632 |
| ALMA$_2$(0.9) | 20.096 ± 0.214 | 3571.05 ± 40.38 | 338.609 |
| PA-I | 21.826 ± 0.239 | 6760.70 ± 47.89 | 4.296 |
| PA-II | 21.478 ± 0.237 | 7068.40 ± 51.32 | 4.536 |
| DUOL | 19.389 ± 0.227 | 7089.85 ± 38.93 | 10.122 |

Table 6: Results on *w7a* (n=24292, d=300).

| Algorithm | Mistake (%) | Support Vectors (#) | Time (s) |
|---|---|---|---|
| Perceptron | 4.027 ± 0.095 | 994.40 ± 23.57 | 1.233 |
| ROMMA | 4.158 ± 0.087 | 1026.75 ± 21.51 | 13.860 |
| agg-ROMMA | 3.500 ± 0.061 | 2317.70 ± 58.92 | 137.975 |
| ALMA$_2$(0.9) | 3.518 ± 0.071 | 1031.05 ± 15.33 | 13.245 |
| PA-I | 3.701 ± 0.057 | 2839.60 ± 41.57 | 3.732 |
| PA-II | 3.571 ± 0.053 | 3391.50 ± 51.94 | 4.719 |
| DUOL | 2.771 ± 0.041 | 1699.80 ± 22.78 | 2.677 |

## 5 Conclusions

This paper presented a novel "double updating" approach to online learning named as "DUOL", which not only updates the weight of the newly added support vector, but also adjusts the weight of one existing support vector that seriously conflicts with the new support vector. We show that the mistake bound for an online classification task can be significantly reduced by the proposed DUOL algorithms. We have conducted an extensive set of experiments by comparing with a number of competing algorithms. Promising empirical results validate the effectiveness of our technique. Future work will address issues of multi-class double updating online learning.

## Acknowledgements

This work was supported in part by MOE tier-1 Grant (RG67/07), NRF IDM Grant (NRF2008IDM-IDM-004-018), National Science Foundation (IIS-0643494), and US Navy Research Office (N00014-09-1-0663).

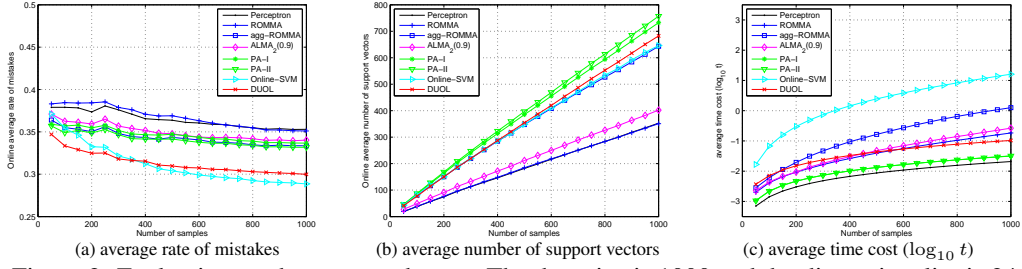

(a) average rate of mistakes     (b) average number of support vectors     (c) average time cost ($\log_{10} t$)

Figure 2: Evaluation on the *german* dataset. The data size is 1000 and the dimensionality is 24.

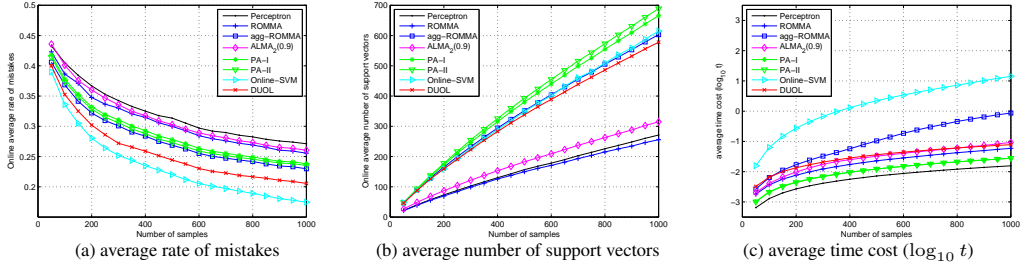

(a) average rate of mistakes     (b) average number of support vectors     (c) average time cost ($\log_{10} t$)

Figure 3: Evaluation on the *splice* dataset. The data size is 1000 and the dimensionality is 60.

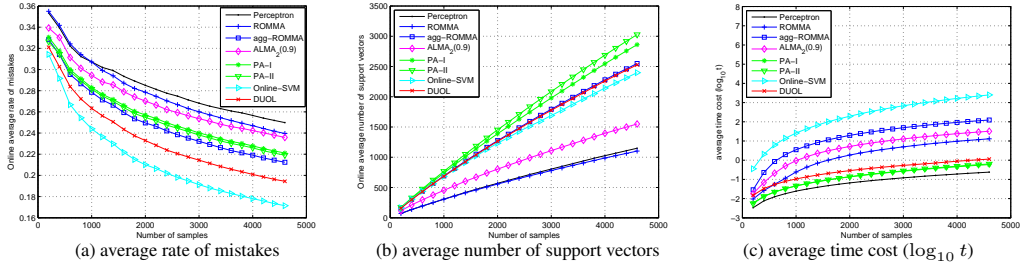

(a) average rate of mistakes     (b) average number of support vectors     (c) average time cost ($\log_{10} t$)

Figure 4: Evaluation on the *spambase* dataset. The data size is 4601 and the dimensionality is 57.

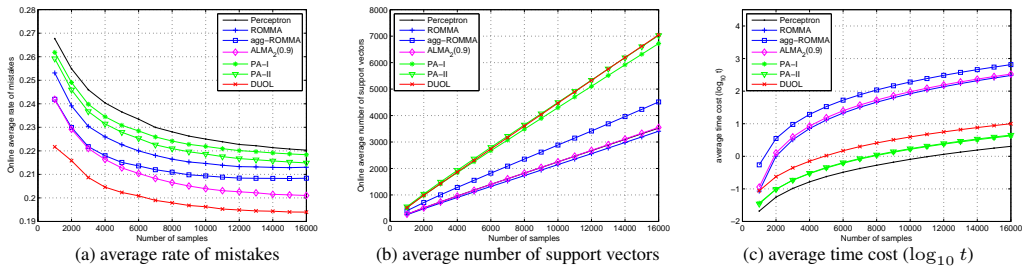

(a) average rate of mistakes     (b) average number of support vectors     (c) average time cost ($\log_{10} t$)

Figure 5: Evaluation on the *a7a* dataset. The data size is 16100 and the dimensionality is 123.

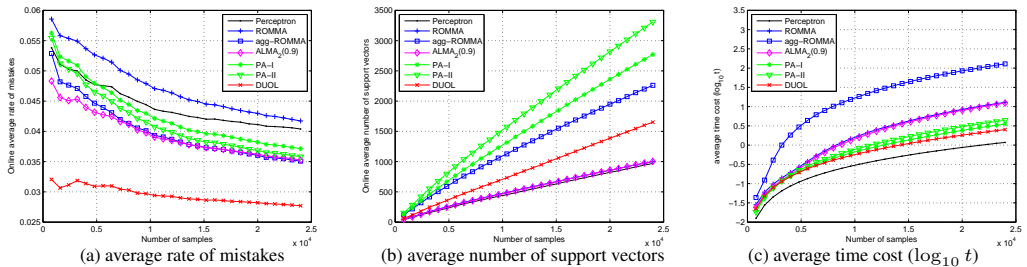

(a) average rate of mistakes     (b) average number of support vectors     (c) average time cost ($\log_{10} t$)

Figure 6: Evaluation on the *w7a* dataset. The data size is 24292 and the dimensionality is 300.

## Footnotes

[1] http://www.csie.ntu.edu.tw/~cjlin/libsvmtools/datasets/

[2] http://www.ics.uci.edu/~mlearn/MLRepository.html

[3] http://cbcl.mit.edu/software-datasets

[4] Due to huge computational cost, we are unable to obtain the results of Online-SVM on two large datasets.

# References

Cavallanti, G., Cesa-Bianchi, N., & Gentile, C. (2007). Tracking the best hyperplane with a simple budget perceptron. *Machine Learning*, *69*, 143–167.

Cesa-Bianchi, N., Conconi, A., & Gentile, C. (2004). On the generalization ability of on-line learning algorithms. *IEEE Trans. on Inf. Theory*, *50*, 2050–2057.

Cesa-Bianchi, N., & Gentile, C. (2006). Tracking the best hyperplane with a simple budget perceptron. *COLT* (pp. 483–498).

Cheng, L., Vishwanathan, S. V. N., Schuurmans, D., Wang, S., & Caelli, T. (2006). Implicit online learning with kernels. *NIPS* (pp. 249–256).

Crammer, K., Dekel, O., Keshet, J., Shalev-Shwartz, S., & Singer, Y. (2006). Online passive-aggressive algorithms. *JMLR*, *7*, 551–585.

Crammer, K., Kandola, J. S., & Singer, Y. (2003). Online classification on a budget. *NIPS*.

Crammer, K., & Singer, Y. (2003). Ultraconservative online algorithms for multiclass problems. *JMLR*, *3*, 951–991.

Dekel, O., Shalev-Shwartz, S., & Singer, Y. (2005). The forgetron: A kernel-based perceptron on a fixed budget. *NIPS*.

Dekel, O., Shalev-Shwartz, S., & Singer, Y. (2008). The forgetron: A kernel-based perceptron on a budget. *SIAM J. Comput.*, *37*, 1342–1372.

Fink, M., Shalev-Shwartz, S., Singer, Y., & Ullman, S. (2006). Online multiclass learning by interclass hypothesis sharing. *ICML* (pp. 313–320).

Freund, Y., & Schapire, R. E. (1999). Large margin classification using the perceptron algorithm. *Mach. Learn.*, *37*, 277–296.

Gentile, C. (2001). A new approximate maximal margin classification algorithm. *JMLR*, *2*, 213–242.

Kivinen, J., Smola, A. J., & Williamson, R. C. (2001a). Online learning with kernels. *NIPS* (pp. 785–792).

Kivinen, J., Smola, A. J., & Williamson, R. C. (2001b). Online learning with kernels. *NIPS* (pp. 785–792).

Li, Y., & Long, P. M. (1999). The relaxed online maximum margin algorithm. *NIPS* (pp. 498–504).

Orabona, F., Keshet, J., & Caputo, B. (2008). The projectron: a bounded kernel-based perceptron. *ICML* (pp. 720–727).

Rosenblatt, F. (1958). The perceptron: A probabilistic model for information storage and organization in the brain. *Psychological Review*, *65*, 386–407.

Shalev-Shwartz, S., & Singer, Y. (2006). Online learning meets optimization in the dual. *COLT* (pp. 423–437).

Weston, J., & Bordes, A. (2005). Online (and offline) on an even tighter budget. *AISTATS* (pp. 413–420).

Yang, L., Jin, R., & Ye, J. (2009). Online learning by ellipsoid method. *ICML* (p. 145).

